# Tight Bounds on Profile Redundancy and Distinguishability

**Jayadev Acharya**
ECE, UCSD
jacharya@ucsd.edu

**Hirakendu Das**
Yahoo!
hdas@yahoo-inc.com

**Alon Orlitsky**
ECE & CSE, UCSD
alon@ucsd.edu

## Abstract

The minimax KL-divergence of any distribution from all distributions in a collection $\mathcal{P}$ has several practical implications. In compression, it is called *redundancy* and represents the least additional number of bits over the entropy needed to encode the output of any distribution in $\mathcal{P}$. In online estimation and learning, it is the lowest expected log-loss regret when guessing a sequence of random values generated by a distribution in $\mathcal{P}$. In hypothesis testing, it upper bounds the largest number of distinguishable distributions in $\mathcal{P}$. Motivated by problems ranging from population estimation to text classification and speech recognition, several machine-learning and information-theory researchers have recently considered label-invariant observations and properties induced by *i.i.d.* distributions. A sufficient statistic for all these properties is the data's *profile*, the multiset of the number of times each data element appears. Improving on a sequence of previous works, we show that the redundancy of the collection of distributions induced over profiles by length-$n$ *i.i.d.* sequences is between $0.3 \cdot n^{1/3}$ and $n^{1/3} \log^2 n$, in particular, establishing its exact growth power.

## 1 Introduction

Information theory, machine learning, and statistics, are closely related disciplines. One of their main intersection areas is the confluence of universal compression, online learning, and hypothesis testing. We consider two concepts in this overlap. The *minimax KL divergence*—a fundamental measure for, among other things, how difficult distributions are to compress, predict, and classify, and *profiles*—a relatively new approach for compression, classification, and property testing over large alphabets. Improving on several previous results, we determine the exact growth power of the KL-divergence minimax of profiles of *i.i.d.* distributions over any alphabet.

### 1.1 Minimax KL divergence

As is well known in information theory, the expected number of bits required to compress data $X$ generated according to a known distribution $P$ is the distribution's entropy, $H(P) = E_P \log 1/P(X)$, and is achieved by encoding $X$ using roughly $\log 1/P(X)$ bits. However, in many applications $P$ is unknown, except that it belongs to a known collection $\mathcal{P}$ of distributions, for example the collection of all *i.i.d.*, or all Markov distributions. This uncertainty typically raises the number of bits above the entropy and is studied in *Universal compression* [9, 13]. Any encoding corresponds to some distribution $Q$ over the encoded symbols. Hence the increase in the expected number of bits used to encode the output of $P$ is $E_P \log 1/Q(X) - H(P) = D(P||Q)$, the *KL divergence* between $P$ and $Q$. Typically one is interested in the highest increase for any distribution $P \in \mathcal{P}$, and finds the encoding that minimizes it. The resulting quantity, called the *(expected) redundancy* of $\mathcal{P}$, *e.g.,* [8, Chap. 13], is therefore the KL minimax

$$\overline{R}(\mathcal{P}) \stackrel{\text{def}}{=} \min_Q \max_{P \in \mathcal{P}} D(P||Q).$$

The same quantity arises in online-learning, *e.g.,* [5, Ch. 9], where the probabilities of random elements $X_1, \ldots, X_n$ are sequentially estimated. One of the most popular measures for the performance of an estimator $Q$ is the per-symbol *log loss* $\frac{1}{n} \sum_{i=1}^{n} \log Q(X_i|X^{i-1})$. As in compression, for underlying distribution $P \in \mathcal{P}$, the expected log loss is $E_P \log 1/Q(X)$, and the *log-loss regret* is $E_P \log 1/Q(X) - H(P) = D(P||Q)$. The maximal expected regret for any distribution in $\mathcal{P}$, minimized over all estimators $Q$ is again the KL minimax, namely, redundancy.

In statistics, redundancy arises in multiple hypothesis testing. Consider the largest number of distributions that can be distinguished from their observations. For example, the largest number of topics distinguishable based on text of a given length. Let $\mathcal{P}$ be a collection of distributions over a support set $\mathcal{X}$. As in [18], a sub-collection $\mathcal{S} \subseteq \mathcal{P}$ of the distributions is $\epsilon$-*distinguishable* if there is a mapping $f : \mathcal{X} \to \mathcal{S}$ such that if $X$ is generated by a distribution $S \in \mathcal{S}$, then $P(f(X) \neq S) \leq \epsilon$. Let $M(\mathcal{P}, \epsilon)$ be the largest number of $\epsilon$-distinguishable distributions in $\mathcal{P}$, and let $h(\epsilon)$ be the binary entropy function. In Section 4 we show that for all $\mathcal{P}$,

$$(1 - \epsilon) \log M(\mathcal{P}, \epsilon) \leq \overline{R}(\mathcal{P}) + h(\epsilon), \tag{1}$$

and in many cases, like the one considered here, the inequality is close to equality.

Redundancy has many other connections to data compression [27, 28], the minimum-description-length principle [3, 16, 17], sequential prediction [21], and gambling [20]. Because of the fundamental nature of $\overline{R}(\mathcal{P})$, and since tight bounds on it often reveal the structure of $\mathcal{P}$, the value of $\overline{R}(\mathcal{P})$ has been studied extensively in all three communities, *e.g.,* the above references as well as [29, 37] and a related minimax in [6].

## 1.2 Redundancy of *i.i.d.* distributions

The most extensively studied collections are independently, identically distributed (*i.i.d.*). For example, for the collection $\mathcal{I}_k^n$ of length-$n$ *i.i.d.* distributions over alphabets of size $k$, a string of works [7, 10, 11, 28, 33, 35, 36] determined the redundancy up to a diminishing additive term,

$$\overline{R}(\mathcal{I}_k^n) = \frac{k-1}{2} \log n + C_k + o(1), \tag{2}$$

where the constant $C_k$ was determined exactly in terms of $k$. For compression this shows that the extra number of bits per symbol required to encode an *i.i.d.* sequence when the underlying distribution is unknown diminishes to zero as $(k-1) \log n / (2n)$. For online learning this shows that these distributions can be learned (or approximated) and that this approximation can be done at the above rate. In hypothesis testing this shows that there are roughly $n^{(k-1)/2}$ distinguishable *i.i.d.* distributions of alphabet size $k$ and length $n$.

Unfortunately, while $\overline{R}(\mathcal{I}_k^n)$ increases logarithmically in the sequence length $n$, it grows linearly in the alphabet size $k$. For sufficiently large $k$, this value even exceeds $n$ itself, showing that general distributions over large alphabets cannot be compressed or learned at a uniform rate over all alphabet sizes, and as the alphabet size increases, progressively larger lengths are needed to achieve a given redundancy, learning rate, or test error.

## 1.3 Patterns

Partly motivated by redundancy's fast increase with the alphabet size, a new approach was recently proposed to address compression, estimation, classification, and property testing over large alphabets.

The *pattern* [25] of a sequence represents the relative order in which its symbols appear. For example, the pattern of abracadabra is 12314151231. A natural method to compress a sequence over a large alphabet is to compress its pattern as well as the *dictionary* that maps the order to the original symbols. For example, for abracadabra, $1 \to a, 2 \to b, 3 \to r, 4 \to c, 5 \to d$.

It can be shown [15, 26] that for all *i.i.d.* distributions, over any alphabet, even infinitely large, as the sequence length increases, essentially all the entropy lies in the pattern, and practically none is in the dictionary. Hence [25] focused on the redundancy of compressing patterns. They showed, *e.g.,* Subsection 1.5, that the although, as in (2), *i.i.d.* sequences over large alphabets have arbitrarily high per-symbol redundancy, and although as above patterns contain essentially all the information of long sequences, the per-symbol redundancy of patterns diminishes to zero at a uniform rate independent of the alphabet size.

In online learning, patterns correspond to estimating the probabilities of each observed symbol, and of all unseen ones combined. For example, after observing the sequence $dad$, with pattern 121, we estimate the probabilities of 1, 2, and 3. The probability we assign to 1 is that of $d$, the probability we assign to 2 is that of $a$, and the probability we assign to 3 is the probability of all remaining letters combined. The aforementioned results imply that while distributions over large alphabets cannot be learned with uniformly diminishing per-symbol log loss, if we would like to estimate the probability of each seen element, but combine together the probabilities of all unseen ones, then the per symbol log loss diminishes to zero uniformly regardless of the alphabet size.

## 1.4 Profiles

Improving on existing pattern-redundancy bounds seems easier to accomplish via *profiles*. Since we consider *i.i.d.* distributions, the order of the elements in a pattern does not affect its probability. For example, for every distribution $P$, $P(112) = P(121)$. It is easy to see that the probability of a pattern is determined by the *fingerprint* [4] or *profile* [25] of the pattern, the multiset of the number of appearances of the symbols in the pattern. For example, the profile of the pattern 121 is $\{1, 2\}$ and all patterns with this profile, 112, 121, 122 will have the same probability under any distribution $P$. Similarly, the profile of 1213 is $\{1, 1, 2\}$ and all patterns with this profile, 1123, 1213, 1231, 1223, 1232, and 1233, will have the same probability under any distribution.

It is easy to see that since all patterns of a given profile have the same probability, the ratio between the actual and estimated probability of a profile is the same as this ratio for each of its patterns. Hence pattern redundancy is the same as profile redundancy [25]. Therefore from now on we consider only profile redundancy, and begin by defining it more formally.

The *multiplicity* $\mu(a)$ of a symbol $a$ in a sequence is the number of times it appears. The *profile* $\overline{\varphi}(\overline{x})$ of a sequence $\overline{x}$ is the multiset of multiplicities of all symbols appearing in it [24, 25]. The profile of the sequence is the multiset of multiplicities. For example, the sequence $ababcde$ has multiplicities $\mu(a) = \mu(b) = 2$, $\mu(c) = \mu(d) = \mu(e) = 1$, and profile $\{1, 1, 1, 2, 2\}$. The *prevalence* $\varphi_\mu$ of a multiplicity $\mu$ is the number of elements with multiplicity $\mu$.

Let $\Phi^n$ denote the collection of all profiles of length-$n$ sequences. For example, for sequences of length one there is a single element appearing once, hence $\Phi^1 = \{\{1\}\}$, for length two, either one element appears twice, or each of two elements appear once, hence $\Phi^2 = \{\{2\}, \{1, 1\}\}$, similarly $\Phi^3 = \{\{3\}, \{2, 1\}, \{1, 1, 1\}\}$, etc.

We consider the distributions induced on $\Phi^n$ by all discrete *i.i.d.* distributions over any alphabet. The probability that an *i.i.d.* distribution $P$ generates an $n$-element sequence $\overline{x}$ is $P(\overline{x}) \overset{\text{def}}{=} \prod_{i=1}^{n} P(x_i)$. The probability of a profile $\overline{\varphi} \in \Phi^n$ is the sum of the probabilities of all sequences of this profile, $P(\overline{\varphi}) \overset{\text{def}}{=} \sum_{\overline{x}:\overline{\varphi}(\overline{x})=\overline{\varphi}} P(\overline{x})$. For example, if $P$ is $B(2/3)$ over $h$ and $t$, then for $n = 3$, $P(\{3\}) = P(hhh) + P(ttt) = 1/3$, $P(\{2, 1\}) = P(hht) + P(hth) + P(thh) + P(tth) + P(tht) + P(htt) = 2/3$, and $P(\{1, 1, 1\}) = 0$ as this $P$ is binary hence at most two symbols can appear. On the other hand, if $P$ is a roll of a fair die, then $P(\{3\}) = 1/36$, $P(\{2, 1\}) = 5/12$, and $P(\{1, 1, 1\}) = 5/9$. We let $\mathcal{I}_\Phi^n = \{P(\overline{\varphi}) : P \text{ is a discrete } i.i.d. \text{ distribution}\}$ be the collection of all distributions on $\Phi^n$ induced by any discrete *i.i.d.* distribution over any alphabet, possibly even infinite.

It is easy to see that any relabeling of the elements in an *i.i.d.* distribution will leave the profile distribution unchanged, for example, if instead of $h$ and $t$ above, we have a distribution over 0's and 1's. Furthermore, profiles are sufficient statistics for every label-invariant property. While many theoretical properties of profiles are known, even calculating the profile probabilities for a given distribution and a profile seems hard [23, 38] in general.

Profile redundancy arises in at least two other machine-learning applications, closeness-testing and classification. In closeness testing [4], we try to determine if two sequences are generated by same or different distributions. In classification, we try to assign a test sequence to one of two training sequences. *Joint profiles* and quantities related to profile redundancy are used to construct competitive closeness tests and classifiers that perform almost as well as the best possible [1, 2].

Profiles also arise in statistics, in estimating *symmetric* or *label-invariant* properties of *i.i.d.* distributions ([34] and references therein). For example the support size, entropy, moments, or number of heavy hitters. All these properties depend only on the multiset of probability values in the distribution. For example, the entropy of the distribution $p(heads) = .6$, $p(tails) = .4$, depends only on the probability multiset $\{.6, .4\}$. For all these properties, profiles are a sufficient statistic.

## 1.5 Previous Results

As patterns and profiles have the same redundancy, we describe the results for profiles.

Instead of the expected redundancy $\overline{R}(\mathcal{I}_\Phi^n)$ that reflects the increase in the expected number of bits, [25] bounded the more stringent but closely-related *worst-case redundancy*, $\hat{R}(\mathcal{I}_\Phi^n)$, reflecting the increase in the worst-case number of bits, namely over all sequences. Using bounds [19] on the partition function, they showed that

$$\Omega(n^{1/3}) \le \hat{R}(\mathcal{I}_\Phi^n) \le \left(\pi\sqrt{\frac{2}{3}}\right) n^{1/2}.$$

These bounds do not involve the alphabet size, hence show that unlike the sequences themselves, patterns (whose redundancy equals that of profiles), though containing essentially all the information of the sequence, can be compressed and learned with redundancy and log-loss diminishing as $n^{-1/2}$, uniformly over all alphabet sizes.

Note however that by contrast to *i.i.d.* distributions, where the redundancy (2) was determined up to a diminishing additive constant, here not even the power was known. Consequently several papers considered improvements of these bounds, mostly for expected redundancy, the minimax KL divergence.

Since expected redundancy is at most the worst-case redundancy, the upper bound applies also for expected redundancy. Subsequently [31] described a partial proof-outline that could potentially show the following tighter upper bound on expected redundancy, and [14] proved the following lower bound, strengthening one in [32],

$$1.84 \left( \frac{n}{\log n} \right)^{1/3} \leq \overline{R}(\mathcal{I}_\Phi^n) \leq n^{0.4}. \tag{3}$$

### 1.6 New results

In Theorem 15 we use error-correcting codes to exhibit a larger class of distinguishable distributions in $\mathcal{I}_\Phi^n$ than was known before, thereby removing the $\log n$ factor from the lower bound in (3). In Theorem 11 we demonstrate a small number of distributions such that every distribution in $\mathcal{I}_\Phi^n$ is within a small KL divergence from one of them, thereby reducing the upper bound to have the same power as the lower bound. Combining these results we obtain,

$$0.3 \cdot n^{1/3} \leq (1 - \epsilon) \log M(\mathcal{I}_\Phi^n, \epsilon) \leq \overline{R}(\mathcal{I}_\Phi^n) \leq n^{1/3} \log^2 n. \tag{4}$$

These results close the power gap between the upper and lower bounds that existed in the literature. They show that when a pattern is compressed or a sequence is estimated (with all unseen elements combined into *new*), the per-symbol redundancy and log-loss decrease to 0 uniformly over all distributions faster than $\log^2 n / n^{2/3}$, a rate that is optimal up to a $\log^2 n$ factor. They also show that for length-$n$ profiles, the redundancy $\overline{R}(\mathcal{I}_\Phi^n)$ is essentially the logarithm $\log M(\mathcal{I}_\Phi^n, \epsilon)$ of the number of distinguishable distributions.

### 1.7 Outline

In the next section we describe properties of Poisson sampling and redundancy that will be used later in the paper. In Section 3 we establish the upper bound and in Section 4, the lower bound. Most of the proofs are provided in the Appendix.

## 2 Preliminaries

We describe some techniques and results used in the proofs.

### 2.1 Poisson sampling

When a distribution is sampled *i.i.d.* exactly $n$ times, the multiplicities are dependent, complicating the analysis of many properties. A standard approach [22] to overcome the dependence is to sample the distribution a random $\text{poi}(n)$ times, the Poisson distribution with parameter $n$, resulting in sequences of random length near close to $n$. We let $\text{poi}(\lambda, \mu) \stackrel{\text{def}}{=} e^{-\lambda} \lambda^\mu / \mu!$ denote the probability that a $\text{poi}(\lambda)$ random variable attains the value $\mu$.

The following basic properties of Poisson sampling help simplify the analysis and relate it to fixed-length sampling.

**Lemma 1.** *If a discrete* i.i.d. *distribution is sampled* $\text{poi}(n)$ *times then: (1) the number of appearances of different symbols are independent; (2) a symbol with probability $p$ appears $\text{poi}(np)$ times; (3) for any fixed $n_0$, conditioned on the length $\text{poi}(n) \geq n_0$, the first $n_0$ elements are distributed identically to sampling $P$ exactly $n_0$ times.*

We now express profile probabilities and redundancy under Poisson sampling. As we saw, the probability of a profile is determined by just the multiset of probability value and the symbol labels are irrelevant. For convenience, we assume that the distribution is over the positive integers, and we replace the distribution parameters $\{p_i\}$ by the Poisson parameters $\{np_i\}$. For a distribution $P = \{p_1, p_2, \ldots\}$, let $\lambda_i \stackrel{\text{def}}{=} np_i$, and $\Lambda = \{\lambda_1, \lambda_2, \ldots\}$. The profile generated

by this distribution is a multiset $\overline{\varphi} = \{\mu_1, \mu_2, \ldots\}$, where each $\mu_i$ generated independently according to $\mathrm{poi}(\lambda_i)$. The probability that $\Lambda$ generates $\overline{\varphi}$ is [1, 25],

$$\Lambda(\overline{\varphi}) = \frac{1}{\prod_{\mu=0}^{\infty} \varphi_\mu!} \sum_\sigma \prod_i \mathrm{poi}(\lambda_{\sigma(i)}, \mu_i). \tag{5}$$

where the summation is over all permutations of the support set.

For example, for $\Lambda = \{\lambda_1, \lambda_2, \lambda_3\}$, the profile $\overline{\varphi} = \{2, 2, 3\}$ can be generated by specifying which element appears three times. This is reflected by the $\varphi_2!$ in the denominator, and each of the repeated terms in the numerator are counted only once.

Similar to $\mathcal{I}_\Phi^n$, we use $\mathcal{I}_\Phi^{\mathrm{poi}(n)}$ to denote the class of distributions induced on $\Phi^* \overset{\Delta}{=} \Phi^0 \cup \Phi^1 \cup \Phi^2 \cup \ldots$ when sequences of length $\mathrm{poi}(n)$ are generated *i.i.d.*. It is easy to see that a distribution in $\mathcal{I}_\Phi^{\mathrm{poi}(n)}$ is a collection of $\lambda_i$'s summing to $n$. The redundancy $\overline{R}(\mathcal{I}_\Phi^{\mathrm{poi}(n)})$, and $\epsilon$-distinguishability $M(\mathcal{I}_\Phi^{\mathrm{poi}(n)}, \epsilon)$ are defined as before. The following lemma shows that bounding $M(\mathcal{I}_\Phi^{\mathrm{poi}(n)}, \epsilon)$ and $\overline{R}(\mathcal{I}_\Phi^{\mathrm{poi}(n)})$ is sufficient to bound $\overline{R}(\mathcal{I}_\Phi^n)$.

**Lemma 2.** *For any fixed $\epsilon > 0$,*

$$(1 - o(1))\overline{R}(\mathcal{I}_\Phi^{n - \sqrt{n}\log n}) \leq \overline{R}(\mathcal{I}_\Phi^{\mathrm{poi}(n)}) \quad and \quad M(\mathcal{I}_\Phi^{\mathrm{poi}(n)}, \epsilon) \leq M(\mathcal{I}_\Phi^{n + \sqrt{n}\log n}, 2\epsilon).$$

*Proof Sketch.* It is easy to show that $\overline{R}(\mathcal{I}_\Phi^n)$ and $M(\mathcal{I}_\Phi^n, \epsilon)$ are non-decreasing in $n$. Combining this with the fact that the probability that $\mathrm{poi}(n)$ is less than $n - \sqrt{n}\log n$ or greater than $n + \sqrt{n}\log n$ goes to 0 yields the bounds. ∎

Finally, the next lemma, proved in the Appendix, provides a simple formula for cross expectations of Poisson distributions.

**Lemma 3.** *For any $\lambda_0, \lambda_1, \lambda_2 > 0$,*

$$\mathbb{E}_{\mu \sim \mathrm{poi}(\lambda_1)} \left[ \frac{\mathrm{poi}(\lambda_2, \mu)}{\mathrm{poi}(\lambda_0, \mu)} \right] = \exp\left( \frac{(\lambda_1 - \lambda_0)(\lambda_2 - \lambda_0)}{\lambda_0} \right).$$

## 2.2 Redundancy

We state some basic properties of redundancy.

For a distribution $P$ over $\mathcal{A}$ and a function $f : \mathcal{A} \to \mathcal{B}$, let $f(P)$ be the distribution over $B$ that assigns to $b \in \mathcal{B}$ the probability $P(f^{-1}(b))$. Similarly, for a collection $\mathcal{P}$ of distributions over $\mathcal{A}$, let $f(\mathcal{P}) = \{f(P) : P \in \mathcal{P}\}$. The convexity of KL-divergence shows that $D(f(P)\|f(Q)) \leq D(P\|Q)$, and can be used to show

**Lemma 4** (Function Redundancy). $\overline{R}(f(\mathcal{P})) \leq \overline{R}(\mathcal{P})$.

For a collection $\mathcal{P}$ of distributions over $\mathcal{A} \times \mathcal{B}$, let $\mathcal{P}_\mathcal{A}$ and $\mathcal{P}_\mathcal{B}$ be the collection of marginal distributions over $\mathcal{A}$ and $\mathcal{B}$, respectively. In general, $\overline{R}(\mathcal{P})$ can be larger or smaller than $\overline{R}(\mathcal{P}_\mathcal{A}) + \overline{R}(\mathcal{P}_\mathcal{B})$. However, when $\mathcal{P}$ consists of *product distributions*, namely $P(a, b) = P_\mathcal{A}(a) \cdot P_\mathcal{B}(b)$, the redundancy of the product is at most the sum of the marginal redundancies. The proof is given in the Appendix.

**Lemma 5** (Redundancy of products). *If $\mathcal{P}$ be a collection of product distributions over $\mathcal{A} \times \mathcal{B}$, then*

$$\overline{R}(\mathcal{P}) \leq \overline{R}(\mathcal{P}_\mathcal{A}) + \overline{R}(\mathcal{P}_\mathcal{B}).$$

For a prefix-free code $C : \mathcal{A} \to \{0, 1\}^*$, let $\mathbb{E}_P[|C|]$ be the expected length of $C$ under distribution $P$. Redundancy is the extra number of bits above the entropy needed to encode the output of any distribution in $\mathcal{P}$. Hence,

**Lemma 6.** *For every prefix-free code $C$,* $\overline{R}(\mathcal{P}) \leq \max_{P \in \mathcal{P}} \mathbb{E}_P[|C|]$.

**Lemma 7** (Redundancy of unions). *If $\mathcal{P}_1, \ldots, \mathcal{P}_T$ are distribution collections, then*

$$\overline{R}\left( \bigcup_{1 \leq i \leq k} \mathcal{P}_i \right) \leq \max_{1 \leq i \leq T} \overline{R}(\mathcal{P}_i) + \log T.$$

# 3  Upper bound

A distribution in $\Lambda \in \mathcal{I}_\Phi^{\mathrm{poi}(n)}$ is a multiset of $\lambda$'s adding to $n$. For any such distribution, let

$$\Lambda_{\mathrm{low}} \overset{\mathrm{def}}{=} \{\lambda \in \Lambda : \lambda \leq n^{1/3}\}, \; \Lambda_{\mathrm{med}} \overset{\mathrm{def}}{=} \{\lambda \in \Lambda : n^{1/3} < \lambda \leq n^{2/3}\}, \; \Lambda_{\mathrm{high}} \overset{\mathrm{def}}{=} \{\lambda \in \Lambda : \lambda > n^{2/3}\},$$

and let $\overline{\varphi}_{\mathrm{low}}, \overline{\varphi}_{\mathrm{med}}, \overline{\varphi}_{\mathrm{high}}$ denote the corresponding profile each subset generates. Then $\overline{\varphi} = \overline{\varphi}_{\mathrm{low}} \cup \overline{\varphi}_{\mathrm{med}} \cup \overline{\varphi}_{\mathrm{high}}$. Let $\mathcal{I}_{\overline{\varphi}_{\mathrm{low}}} = \{\Lambda_{\mathrm{low}} : \Lambda \in \mathcal{I}_\Phi^{\mathrm{poi}(n)}\}$ be the collection of all $\Lambda_{\mathrm{low}}$. Note that $n$ is implicit here and in the rest of the paper. A distribution in $\mathcal{I}_{\overline{\varphi}_{\mathrm{low}}}$ is a multiset of $\lambda$'s such that each is $\leq n^{1/3}$ and they sum to either $n$ or to $\leq n - n^{1/3}$. $\mathcal{I}_{\overline{\varphi}_{\mathrm{med}}}$ and $\mathcal{I}_{\overline{\varphi}_{\mathrm{high}}}$ are defined similarly.

$\overline{\varphi}$ is determined by the triple $(\overline{\varphi}_{\mathrm{low}}, \overline{\varphi}_{\mathrm{med}}, \overline{\varphi}_{\mathrm{high}})$, and by Poisson sampling, $\overline{\varphi}_{\mathrm{low}}, \overline{\varphi}_{\mathrm{med}}$ and $\overline{\varphi}_{\mathrm{high}}$ are independent. Hence by Lemmas 4 and 5,

$$\overline{R}(\mathcal{I}_\Phi^n) \leq \overline{R}(\mathcal{I}_{(\overline{\varphi}_{\mathrm{low}}, \overline{\varphi}_{\mathrm{med}}, \overline{\varphi}_{\mathrm{high}})}) \leq \overline{R}(\mathcal{I}_{\overline{\varphi}_{\mathrm{low}}}) + \overline{R}(\mathcal{I}_{\overline{\varphi}_{\mathrm{med}}}) + \overline{R}(\mathcal{I}_{\overline{\varphi}_{\mathrm{high}}}).$$

In Subsection 3.1 we show that $\mathcal{I}_{\overline{\varphi}_{\mathrm{low}}} < 4n^{1/3} \log n$ and $\mathcal{I}_{\overline{\varphi}_{\mathrm{high}}} < 4n^{1/3} \log n$. In Subsection 3.2 we show that $\mathcal{I}_{\overline{\varphi}_{\mathrm{med}}} < \frac{1}{2}n^{1/3} \log^2 n$.

In the next two subsections we elaborate on the overview and sketch some proof details.

## 3.1  Bounds on $\overline{R}(\mathcal{I}_{\overline{\varphi}_{\mathrm{low}}})$ and $\overline{R}(\mathcal{I}_{\overline{\varphi}_{\mathrm{high}}})$

Elias Codes [12] are prefix-free codes that encode a positive integer $n$ using at most $\log n + \log(\log n + 1) + 1$ bits. We use Elias codes and design explicit coding schemes for distributions in $\mathcal{I}_{\overline{\varphi}_{\mathrm{low}}}$ and $\mathcal{I}_{\overline{\varphi}_{\mathrm{high}}}$, and prove the following result.

**Lemma 8.** $\overline{R}(\mathcal{I}_{\overline{\varphi}_{\mathrm{low}}}) < 4n^{1/3} \log n$, and $\overline{R}(\mathcal{I}_{\overline{\varphi}_{\mathrm{high}}}) < 2n^{1/3} \log n$.

*Proof.* Any distribution $\Lambda_{\mathrm{high}} \in \mathcal{I}_{\overline{\varphi}_{\mathrm{high}}}$ consists of $\lambda$'s that are $> n^{2/3}$ and add to $\leq n$. Hence $|\Lambda_{\mathrm{high}}|$ is $< n^{1/3}$, and so is the number of multiplicities in $\overline{\varphi}_{\mathrm{high}}$. Each multiplicity is a $\mathrm{poi}(\lambda)$ random variable, and is encoded separately using Elias code. For example, the profile $\{100, 100, 200, 250, 500\}$ is encoded by coding the sequence 100, 100, 200, 250, 500 all using Elias scheme. For $\lambda > 10$, the number of bits needed to encode a $\mathrm{poi}(\lambda)$ random variable using Elias codes can be shown to be at most $2 \log \lambda$. The expected code-length is at most $n^{1/3} \cdot 2 \log n$. Applying Lemma 6 gives $\overline{R}(\mathcal{I}_{\overline{\varphi}_{\mathrm{high}}}) < 2n^{1/3} \log n$.

A distribution $\Lambda_{\mathrm{low}} \in \mathcal{I}_{\overline{\varphi}_{\mathrm{low}}}$ consists of $\lambda$'s less that $< n^{1/3}$ and sum at most $n$. We encode distinct multiplicities along with their prevalences, using two integers for each distinct multiplicity. For example, $\overline{\varphi} = \{1, 1, 1, 1, 1, 2, 2, 2, 5\}$ is coded as $1, 5, 2, 3, 5, 1$. Using Poisson tail bounds, we bound the largest multiplicity in $\overline{\varphi}_{\mathrm{low}}$, and use arguments similar to $\mathcal{I}_{\overline{\varphi}_{\mathrm{high}}}$ to obtain $\overline{R}(\mathcal{I}_{\overline{\varphi}_{\mathrm{low}}}) < 4n^{1/3} \log n$. $\blacksquare$

## 3.2  Bound on $\overline{R}(\mathcal{I}_{\overline{\varphi}_{\mathrm{med}}})$

We partition the interval $(n^{1/3}, n^{2/3}]$ into $B = n^{1/3}$ bins. For each distribution in $\mathcal{I}_{\overline{\varphi}_{\mathrm{med}}}$, we divide the $\lambda$'s in it according to these bins. We show that within each interval, there is a uniform distribution such that the KL divergence between the underlying distribution and the induced uniform distribution is *small*. We then show that the number of uniform distributions needed is at most $\exp(n^{1/3} \log n)$. We expand on these ideas and bound $\overline{R}(\mathcal{I}_{\overline{\varphi}_{\mathrm{med}}})$.

We partition $\mathcal{I}_{\overline{\varphi}_{\mathrm{med}}}$ into $T \leq \exp(n^{1/3} \log n)$ classes, upper bound the redundancy of each class, and then invoke Lemma 7 to obtain an upper bound on $\overline{R}(\mathcal{I}_{\overline{\varphi}_{\mathrm{med}}})$. A distribution $\Lambda = \{\lambda_1, \lambda_2, \dots, \lambda_r\} \in \mathcal{I}_{\overline{\varphi}_{\mathrm{med}}}$ is such that $\lambda_i \in [n^{1/3}, n^{2/3}]$ and $\sum_{i=1}^r \lambda_i \leq n$.

Consider any partition of $(n^{1/3}, n^{2/3}]$ into $B \overset{\mathrm{def}}{=} n^{1/3}$ consecutive intervals $I_1, I_2, \dots, I_B$ of lengths $\Delta_1, \Delta_2, \dots, \Delta_B$. For each distribution $\Lambda \in \mathcal{I}_{\overline{\varphi}_{\mathrm{med}}}$, let $\Lambda_j \overset{\mathrm{def}}{=} \{\lambda_{j,l} : l = 1, 2, \dots, m_j\} \overset{\mathrm{def}}{=} \{\lambda : \lambda \in \Lambda \cap I_j\}$ be the set of elements of $\Lambda$ in $I_j$ where $m_j \overset{\mathrm{def}}{=} m_j(\Lambda) \overset{\mathrm{def}}{=} |\Lambda_j|$ is the number of elements of $\Lambda$ in $I_j$. Let

$$\tau(\Lambda) \overset{\mathrm{def}}{=} (m_1, m_2, \dots, m_B)$$

be the $B$-tuple of the counts of $\lambda$'s in each interval.

For example, if $n = 1000$, then $n^{1/3} = 10$ and $n^{2/3} = 100$. For simplicity, we choose $B = 3$ instead of $n^{1/3}$ and $\Delta_1 = 10, \Delta_2 = 30, \Delta_3 = 50$, so the intervals are $I_1 = (10, 20], I_2 = (20, 50], I_3 = (50, 100]$. Suppose, $\Lambda = \{12, 15, 25, 35, 32, 43, 46, 73\}$, then $\Lambda_1 = \{12, 15\}, \Lambda_2 = \{25, 35, 32, 43, 46\}, \Lambda_3 = \{73\}$ and $\tau(\Lambda) = (m_1, m_2, m_3) = (2, 5, 1)$.

We partition $\mathcal{I}_{\overline{\varphi}_{\mathrm{med}}}$, such that two distributions $\Lambda$ and $\Lambda'$ are in the same class if and only if $\tau(\Lambda) = \tau(\Lambda')$. Thus each class of distributions is characterized by a $B$-tuple of integers $\tau = (m_1, m_2, \ldots, m_B)$ and let $\mathcal{I}_\tau$ denote this class. Let $\mathcal{T} \stackrel{\mathrm{def}}{=} \mathcal{T}(\overline{\Delta})$ be the set of all possible different $\tau$ (such that $\mathcal{I}_\tau$ is non-empty), and $T = |\mathcal{T}|$ be the number of classes. We first bound $T$ below. Observe that for any $\Lambda \in \mathcal{I}_{\overline{\varphi}_{\mathrm{med}}}$, and any $j$, we have $m_j < n^{2/3}$, otherwise $\sum_{\lambda \in \Lambda} \lambda > m_j \cdot n^{1/3} = n$. So, each $m_j$ in $\tau$ can take at most $n^{2/3} < n$ values. So, $T < (n^{2/3})^B < n^{n^{1/3}} = \exp(n^{1/3} \log n)$.

For any choice of $\overline{\Delta}$, let $\lambda_j^- \stackrel{\mathrm{def}}{=} n^{1/3} + \sum_{i=1}^{j-1} \Delta_i$ be the left end point of the interval $I_j$ for $j = 1, 2, \ldots, B$. We upper bound $\overline{R}(\mathcal{I}_\tau)$ of any particular class $\tau = (m_1, m_2, \ldots, m_B)$ in the following result.

**Lemma 9.** *For all choices of $\overline{\Delta} = (\Delta_1, \ldots, \Delta_B)$, and all classes $\mathcal{I}_\tau$ such that $\tau = (m_1, \ldots, m_B) \in \mathcal{T}(\overline{\Delta})$,*

$$\overline{R}(\mathcal{I}_\tau) \leq \sum_{j=1}^{B} m_j \frac{\Delta_j^2}{\lambda_j^-}.$$

*Proof Sketch.* For any choice of $\overline{\Delta}$, $\tau = (m_1, \ldots, m_B) \in \mathcal{T}(\overline{\Delta})$, we show a distribution $\Lambda^* \in \mathcal{I}_\tau$ such that for all $\Lambda \in \mathcal{I}_\tau$, $D(\Lambda || \Lambda^*) \leq \sum_{j=1}^{B} m_j \frac{\Delta_j^2}{\lambda_j^-}$. Recall that for $\Lambda \in \mathcal{I}_\tau$, $\Lambda_j$ is the set of elements of $\Lambda$ in $I_j$. Let $\overline{\varphi}_j$ be the profile generated by $\Lambda_j$. Then, $\overline{\varphi}_{\mathrm{med}} = \overline{\varphi}_1 \cup \ldots \cup \overline{\varphi}_B$. The distribution $\Lambda^*$ is chosen to be of the form $\{\lambda_1^* \times m_1, \lambda_2^* \times m_2, \ldots, \lambda_B^* \times m_B\}$, *i.e.*, each $\Lambda_j^*$ is uniform. The result follows from Lemma 3, and the details are in the Appendix . ∎

We now prove that $\overline{R}(\mathcal{I}_{\overline{\varphi}_{\mathrm{med}}}) < \frac{1}{2} n^{1/3} \log^2 n$.

By Lemma 7 it suffices to bound $\overline{R}(\mathcal{I}_\tau)$. From Theorem 9 it follows that the choice of $\overline{\Delta}$ determines the bound on $\overline{R}(\mathcal{I}_\tau)$. A solution to the following optimization problem yields a bound :

$$\min_{\overline{\Delta}} \max_\tau \sum_{j=1}^{B} m_j \frac{\Delta_j^2}{\lambda_j^-}, \text{ subject to } \sum_{j=1}^{B} m_j \lambda_j^- \leq n.$$

Instead of minimizing over all partitions, we choose the endpoints of the intervals as a geometric series as a bound for the expression. The left-end point of $I_j$ is $\lambda_j^-$, so $\lambda_1^- = n^{1/3}$. We let $\lambda_{j+1}^- = \lambda_j^- (1 + c)$. The constant $c$ is chosen to ensure that $\lambda_1^- (1+c)^B = n^{1/3} (1+c)^{n^{1/3}} = n^{2/3}$, the right end-point of $I_B$. This yields, $c < 2 \log(1+c) = \frac{2 \log(n^{1/3})}{n^{1/3}}$. Now, $\Delta_j = \lambda_{j+1}^- - \lambda_j^- = c \lambda_j^-$, so $\frac{\Delta_j^2}{\lambda_j^-} = c^2 \lambda_j^-$. This translates the objective function to the constraint, and is in fact the optimal intervals for the optimization problem (details omitted). Using this, for any $\tau = (m_1, \ldots, m_B) \in \mathcal{T}(\overline{\Delta})$,

$$\sum_{j=1}^{B} m_j \frac{\Delta_j^2}{\lambda_j^-} = c^2 \sum_{j=1}^{B} m_j \lambda_j^- \leq c^2 n < \left(\frac{2 \log(n^{1/3})}{n^{1/3}}\right)^2 n = \frac{4}{9} n^{1/3} \log^2 n.$$

This, along with Lemma 7 gives the following Corollary for sufficiently large $n$.

**Corollary 10.** *For large $n$, $\overline{R}(\mathcal{I}_{\overline{\varphi}_{\mathrm{med}}}) < \frac{1}{2} \cdot n^{1/3} \log^2 n$.*

Combining Lemma 8 with this result yields,

**Theorem 11.** *For sufficiently large $n$,*

$$\overline{R}(\mathcal{I}_\Phi^n) \leq n^{1/3} \log^2 n.$$

## 4 Lower bound

We use error-correcting codes to construct a collection of $2^{0.3n^{1/3}}$ distinguishable distributions, improving by a logarithmic factor the bound in [14, 31].

The convexity of KL-divergence can be used to show

**Lemma 12.** *Let $P$ and $Q$ be distributions on $\mathcal{A}$. Suppose $\mathcal{A}_1 \subset \mathcal{A}$ be such that $P(\mathcal{A}_1) \geq 1 - \epsilon > 1/2$, $Q(\mathcal{A}_1) \leq \delta < 1/2$. Then, $D(P||Q) \geq (1-\epsilon) \log\left(\frac{1}{\delta}\right) - h(\epsilon)$.*

We use this result to show that $(1 - \epsilon) \log M(\mathcal{P}, \epsilon) \leq \overline{R}(\mathcal{P})$. Recall that for $\mathcal{P}$ over $\mathcal{A}$, $M \overset{\text{def}}{=} M(\mathcal{P}, \epsilon)$ is the largest number of $\epsilon-$distinguishable distributions in $\mathcal{P}$. Let $P_1, P_2, \ldots, P_M$ in $\mathcal{P}$ and $\mathcal{A}_1, \mathcal{A}_2, \ldots, \mathcal{A}_M$ be a partition of $\mathcal{A}$ such that $P_j(\mathcal{A}_j) \geq 1-\epsilon$. Let $Q_0$ be the distribution such that, $\overline{R}(\mathcal{P}) = \sup_{P \in \mathcal{P}} D(P||Q_0)$. Since $\sum_{j=1}^{M} Q_0(\mathcal{A}_j) = 1$, $Q_0(\mathcal{A}_m) < \frac{1}{M}$ for some $m \in \{1, \ldots, M\}$. Also, $P_m(\mathcal{A}_m) \geq 1 - \epsilon$. Plugging in $P = P_m$, $Q = Q_0$, $\mathcal{A}_1 = \mathcal{A}_m$, and $\delta = 1/M$ in the Lemma 12,

$$\overline{R}(\mathcal{P}) \geq D(P_m||Q_0) \geq (1 - \epsilon) \log\left(M(\mathcal{P}, \epsilon)\right) - h(\epsilon).$$

We now describe the class of distinguishable distributions. Fix $C > 0$. Let $\lambda_i^* \overset{\text{def}}{=} Ci^2$, $K \overset{\text{def}}{=} \lfloor (3n/C)^{1/3} \rfloor$, and $\mathcal{S} \overset{\text{def}}{=} \{\lambda_i^* : 1 \leq i \leq K\}$. $K$ is chosen so that sum of elements in $\mathcal{S}$ is at most $n$. Let $\overline{x} = x_1 x_2 \ldots x_K$ be a binary string and

$$\Lambda_{\overline{x}} \overset{\text{def}}{=} \{\lambda_i^* : x_i = 1\} \cup \left\{ n - \sum \lambda_i^* x_i \right\}.$$

The distribution contains $\lambda_i^*$ whenever $x_i = 1$, and the last element ensures that the elements add up to $n$. A *binary code* of length $k$ and minimum distance $d_{\min}$ is a collection of $k-$length binary strings with Hamming distance between any two strings is at least $d_{\min}$. The size of the code is the number of elements (codewords) in it. The following shows the existence of codes with a specified minimum distance and size.

**Lemma 13** ([30]). *Let $\frac{1}{2} > \alpha > 0$. There exists a code with $d_{\min} \geq \alpha k$ and size $\geq 2^{k(1-h(\alpha)-o(1))}$.*

Let $\mathcal{C}$ be a code satisfying Lemma 13 for $k = K$ and let $\mathcal{L} = \{\Lambda_{\overline{c}} : \overline{c} \in \mathcal{C}\}$ be the set of distributions generated by using the strings in $\mathcal{C}$. The following result shows that distributions in $\mathcal{L}$ are distinguishable and is proved in Appendix .

**Lemma 14.** *The set $\mathcal{L}$ is $\frac{2e^{-C/4}}{\alpha}-$distinguishable.*

Plugging $\alpha = 5 \times 10^{-5}$ and $C = 60$, then Lemma 13 and Equation (1) yields,

**Theorem 15.** *For sufficiently large $n$,*

$$0.3 \cdot n^{1/3} \leq \overline{R}(\mathcal{I}_\Phi^n).$$

## Acknowledgments

The authors thank Ashkan Jafarpour and Ananda Theertha Suresh for many helpful discussions.

## References

[1] J. Acharya, H. Das, A. Jafarpour, A. Orlitsky, and S. Pan. Competitive closeness testing. *J. of Machine Learning Research - Proceedings Track*, 19:47–68, 2011.

[2] J. Acharya, H. Das, A. Jafarpour, A. Orlitsky, S. Pan, and A. T. Suresh. Competitive classification and closeness testing. *Journal of Machine Learning Research - Proceedings Track*, 23:22.1–22.18, 2012.

[3] A. R. Barron, J. Rissanen, and B. Yu. The minimum description length principle in coding and modeling. *IEEE Transactions on Information Theory*, 44(6):2743–2760, 1998.

[4] T. Batu, L. Fortnow, R. Rubinfeld, W. D. Smith, and P. White. Testing that distributions are close. In *Annual Symposium on Foundations of Computer Science*, page 259, 2000.

[5] N. Cesa-Bianchi and G. Lugosi. *Prediction, Learning, and Games*. Cambridge University Press, New York, NY, USA, 2006.

[6] K. Chaudhuri and A. McGregor. Finding metric structure in information theoretic clustering. In *Conference on Learning Theory*, pages 391–402, 2008.

[7] T. Cover. Universal portfolios. *Mathematical Finance*, 1(1):1–29, January 1991.

[8] T. Cover and J. Thomas. *Elements of Information Theory, 2nd Ed.* Wiley Interscience, 2006.

[9] L. Davisson. Universal noiseless coding. *IEEE Transactions on Information Theory*, 19(6):783–795, November 1973.

[10] L. D. Davisson, R. J. McEliece, M. B. Pursley, and M. S. Wallace. Efficient universal noiseless source codes. *IEEE Transactions on Information Theory*, 27(3):269–279, 1981.

[11] M. Drmota and W. Szpankowski. Precise minimax redundancy and regret. *IEEE Transactions on Information Theory*, 50(11):2686–2707, 2004.

[12] P. Elias. Universal codeword sets and representations of the integers. *IEEE Transactions on Information Theory*, 21(2):194–203, Mar 1975.

[13] B. M. Fitingof. Optimal coding in the case of unknown and changing message statistics. *Probl. Inform. Transm.*, 2(2):1–7, 1966.

[14] A. Garivier. A lower-bound for the maximin redundancy in pattern coding. *Entropy*, 11(4):634–642, 2009.

[15] G. M. Gemelos and T. Weissman. On the entropy rate of pattern processes. *IEEE Transactions on Information Theory*, 52(9):3994–4007, 2006.

[16] P. Grünwald. A tutorial introduction to the minimum description length principle. *CoRR*, math.ST/0406077, 2004.

[17] P. Grünwald, J. S. Jones, J. de Winter, and É. Smith. Safe learning: bridging the gap between bayes, mdl and statistical learning theory via empirical convexity. *J. of Machine Learning Research - Proceedings Track*, 19:397–420, 2011.

[18] P. D. Grünwald. *The Minimum Description Length Principle*. The MIT Press, 2007.

[19] G. Hardy and S. Ramanujan. Asymptotic formulae in combinatory analysis. *Proceedings of London Mathematics Society*, 17(2):75–115, 1918.

[20] J. Kelly. A new interpretation of information rate. *IEEE Transactions on Information Theory*, 2(3):185–189, 1956.

[21] N. Merhav and M. Feder. Universal prediction. *IEEE Transactions on Information Theory*, 44(6):2124–2147, October 1998.

[22] M. Mitzenmacher and E. Upfal. *Probability and computing - randomized algorithms and probabilistic analysis*. Cambridge University Press, 2005.

[23] A. Orlitsky, S. Pan, Sajama, N. Santhanam, and K. Viswanathan. Pattern maximum likelihood: computation and experiments. *In preparation*, 2012.

[24] A. Orlitsky, N. Santhanam, K. Viswanathan, and J. Zhang. On modeling profiles instead of values. In *Proceedings of the 20th conference on Uncertainty in artificial intelligence*, 2004.

[25] A. Orlitsky, N. Santhanam, and J. Zhang. Universal compression of memoryless sources over unknown alphabets. *IEEE Transactions on Information Theory*, 50(7):1469– 1481, July 2004.

[26] A. Orlitsky, N. P. Santhanam, K. Viswanathan, and J. Zhang. Limit results on pattern entropy. *IEEE Transactions on Information Theory*, 52(7):2954–2964, 2006.

[27] J. Rissanen. Universal coding, information, prediction, and estimation. *IEEE Transactions on Information Theory*, 30(4):629–636, July 1984.

[28] J. Rissanen. Fisher information and stochastic complexity. *IEEE Transactions on Information Theory*, 42(1):40–47, January 1996.

[29] J. Rissanen, T. P. Speed, and B. Yu. Density estimation by stochastic complexity. *IEEE Transactions on Information Theory*, 38(2):315–323, 1992.

[30] R. M. Roth. *Introduction to coding theory*. Cambridge University Press, 2006.

[31] G. Shamir. A new upper bound on the redundancy of unknown alphabets. In *Proceedings of The 38th Annual Conference on Information Sciences and Systems, Princeton, New-Jersey*, 2004.

[32] G. Shamir. Universal lossless compression with unknown alphabets—the average case. *IEEE Transactions on Information Theory*, 52(11):4915–4944, November 2006.

[33] W. Szpankowski. On asymptotics of certain recurrences arising in universal coding. *Problems of Information Transmission*, 34(2):142–146, 1998.

[34] P. Valiant. *Testing symmetric properties of distributions*. PhD thesis, Cambridge, MA, USA, 2008. AAI0821026.

[35] F. M. J. Willems, Y. M. Shtarkov, and T. J. Tjalkens. The context-tree weighting method: basic properties. *IEEE Transactions on Information Theory*, 41(3):653–664, 1995.

[36] Q. Xie and A. Barron. Asymptotic minimax regret for data compression, gambling and prediction. *IEEE Transactions on Information Theory*, 46(2):431–445, March 2000.

[37] B. Yu and T. P. Speed. A rate of convergence result for a universal d-semifaithful code. *IEEE Transactions on Information Theory*, 39(3):813–820, 1993.

[38] J. Zhang. *Universal Compression and Probability Estimation with Unknown Alphabets*. PhD thesis, UCSD, 2005.

